# Familiarity Discrimination of Radar Pulses

Eric Granger[1], Stephen Grossberg[2]
Mark A. Rubin[2], William W. Streilein[2]

[1]Department of Electrical and Computer Engineering
École Polytechnique de Montréal
Montreal, Qc. H3C 3A7 CANADA

[2]Department of Cognitive and Neural Systems, Boston University
Boston, MA 02215 USA

## Abstract

The ARTMAP-FD neural network performs both identification (placing test patterns in classes encountered during training) and familiarity discrimination (judging whether a test pattern belongs to any of the classes encountered during training). The performance of ARTMAP-FD is tested on radar pulse data obtained in the field, and compared to that of the nearest-neighbor-based NEN algorithm and to a $k > 1$ extension of NEN.

## 1   Introduction

The recognition process involves both identification and familiarity discrimination. Consider, for example, a neural network designed to identify aircraft based on their radar reflections and trained on sample reflections from ten types of aircraft $A \dots J$. After training, the network should correctly classify radar reflections belonging to the familiar classes $A \dots J$, but it should also abstain from making a meaningless guess when presented with a radar reflection from an object belonging to a different, unfamiliar class. Familiarity discrimination is also referred to as "novelty detection," a "reject option," and "recognition in partially exposed environments."

ARTMAP-FD, an extension of fuzzy ARTMAP that performs familiarity discrimination, has shown its effectiveness on datasets consisting of simulated radar range profiles from aircraft targets [1, 2]. In the present paper we examine the performance of ARTMAP-FD on radar pulse data obtained in the field, and compare it

to that of NEN, a nearest-neighbor-based familiarity discrimination algorithm, and to a $k > 1$ extension of NEN.

## 2  Fuzzy ARTMAP

Fuzzy ARTMAP [3] is a self-organizing neural network for learning, recognition, and prediction. Each input $\mathbf{a}$ learns to predict an output class $K$. During training, the network creates internal recognition categories, with the number of categories determined on-line by predictive success. Components of the vector $\mathbf{a}$ are scaled so that each $a_i \in [0, 1]$ $(i = 1 \ldots M)$. Complement coding [4] doubles the number of components in the input vector, which becomes $\mathbf{A} \equiv (\mathbf{a}, \mathbf{a}^c)$, where the $i^{th}$ component of $\mathbf{a}^c$ is $a_i^c \equiv (1 - a_i)$. With fast learning, the weight vector $\mathbf{w}_j$ records the largest and smallest component values of input vectors placed in the $j^{th}$ category. The $2M$-dimensional vector $\mathbf{w}_j$ may be visualized as the hyperbox $R_j$ that just encloses all the vectors $\mathbf{a}$ that selected category $j$ during training.

Activation of the coding field $F_2$ is determined by the Weber law choice function $T_j(\mathbf{A}) = |\mathbf{A} \wedge \mathbf{w}_j| / (\alpha + |\mathbf{w}_j|)$, where $(\mathbf{P} \wedge \mathbf{Q})_i \equiv \min(P_i, Q_i)$ and $|\mathbf{P}| \equiv \sum_{i=1}^{2M} |P_i|$. With winner-take-all coding, the $F_2$ node $J$ that receives the largest $F_1 \to F_2$ input $T_j$ becomes active. Node $J$ remains active if it satisfies the matching criterion: $|\mathbf{A} \wedge \mathbf{w}_j| / |\mathbf{A}| = |\mathbf{A} \wedge \mathbf{w}_j| / M > \rho$, where $\rho \in [0, 1]$ is the dimensionless *vigilance parameter*. Otherwise, the network resets the active $F_2$ node and searches until $J$ satisfies the matching criterion. If node $J$ then makes an incorrect class prediction, a *match tracking* signal raises vigilance just enough to induce a search, which continues until either some $F_2$ node becomes active for the first time, in which case $J$ learns the correct output class label $k(J) = K$; or a node $J$ that has previously learned to predict $K$ becomes active. During testing, a pattern $\mathbf{a}$ that activates node $J$ is predicted to belong to the class $K = k(J)$.

## 3  ARTMAP-FD

**Familiarity measure.** During testing, an input pattern $\mathbf{a}$ is defined as *familiar* when a familiarity function $\phi(\mathbf{A})$ is greater than a decision threshold $\gamma$. Once a category choice has been made by the winner-take-all rule, fuzzy ARTMAP ignores the size of the input $T_J$. In contrast, ARTMAP-FD uses $T_J$ to define familiarity, taking

$$\phi(\mathbf{A}) = \frac{T_J(\mathbf{A})}{T_J^{MAX}} = \frac{|\mathbf{A} \wedge \mathbf{w}_J|}{|\mathbf{w}_J|}, \tag{1}$$

where $T_J^{MAX} = |\mathbf{w}_J| / (\alpha + |\mathbf{w}_J|)$. This maximal value of $T_J$ is attained by each input $\mathbf{a}$ that lies in the hyperbox $R_J$, since $|\mathbf{A} \wedge \mathbf{w}_J| = |\mathbf{w}_J|$ for these points. An input that chooses category $J$ during testing is then assigned the maximum familiarity value 1 if and only if $\mathbf{a}$ lies within $R_J$.

**Familiarity discrimination algorithm.** ARTMAP-FD is identical to fuzzy ARTMAP during training. During testing, $\phi(\mathbf{A})$ is computed after fuzzy ARTMAP has yielded a winning node $J$ and a predicted class $K = k(J)$. If $\phi(\mathbf{A}) > \gamma$, ARTMAP-FD predicts class $K$ for the input $\mathbf{a}$. If $\phi(\mathbf{A}) \leq \gamma$, $\mathbf{a}$ is regarded as belonging to an unfamiliar class and the network makes no prediction.

Note that fuzzy ARTMAP can also abstain from classification, when the baseline vigilance parameter $\bar{\rho}$ is greater than zero during testing. Typically $\bar{\rho} = 0$ during training, to maximize code compression. In radar range profile simulations such

as those described below, fuzzy ARTMAP can perform familiarity discrimination when $\bar{\rho} > 0$ during both training and testing. However, accurate discrimination requires that $\bar{\rho}$ be close to 1, which causes category proliferation during training.

Range profile simulations have also set $\bar{\rho} = 0$ during both training and testing, but with the familiarity measure set equal to the fuzzy ARTMAP match function:

$$\phi(\mathbf{A}) = \frac{|\mathbf{A} \wedge \mathbf{w}_J|}{M}. \tag{2}$$

This approach is essentially equivalent to taking $\bar{\rho} = 0$ during training and $\bar{\rho} > 0$ during testing, with $\bar{\rho} = \gamma$. However, for a test set input $\mathbf{a} \in R_J$, the function defined by (2) sets $\phi(\mathbf{A}) = |\mathbf{w}_J|/M$, which may be large or small although $\mathbf{a}$ is familiar. Thus this function does not provide as good familiarity discrimination as the one defined by (1), which always sets $\phi(\mathbf{A}) = 1$ when $\mathbf{a} \in R_J$. Except as noted, all the simulations below employ the function (1), with $\bar{\rho} = 0$.

**Sequential evidence accumulation.** ART-EMAP (Stage 3) [5] identifies a test set object's class after exposure to a sequence of input patterns, such as differing views, all identified with that one object. Training is identical to that of fuzzy ARTMAP, with winner-take-all coding at $F_2$. ART-EMAP generally employs distributed $F_2$ coding during testing. With winner-take-all coding during testing as well as training, ART-EMAP predicts the object's class to be the one selected by the largest number of inputs in the sequence. Extending this approach, ARTMAP-FD accumulates familiarity measures for each predicted class $K$ as the test set sequence is presented. Once the winning class is determined, the object's familiarity is defined as the average accumulated familiarity measure of the predicted class during the test sequence.

## 4 Familiarity discrimination simulations

Since familiarity discrimination involves placing an input into one of two sets, familiar and unfamiliar, the receiver operating characteristic (ROC) formalism can be used to evaluate the effectiveness of ARTMAP-FD on this task. The *hit rate* $H$ is the fraction of familiar targets the network correctly identifies as familiar and the *false alarm rate* $F$ is the fraction of unfamiliar targets the network incorrectly identifies as familiar. An ROC curve is a plot of $H$ *vs.* $F$, parameterized by the threshold $\gamma$ (*i.e.*, it is equivalent to the two curves $F(\gamma)$ and $H(\gamma)$). The area under the ROC curve is the *c-index*, a measure of predictive accuracy that is independent of both the fraction of positive (familiar) cases in the test set and the positive-case decision threshold $\gamma$.

An ARTMAP-FD network was trained on simulated radar range profiles from 18 targets out of a 36-target set (Fig. 1a). Simulations tested sequential evidence accumulation performance for 1, 3, and 100 observations, corresponding to 0.05, 0.15, and 5.0 sec. of observation (smooth curves, Fig. 1b). As in the case of identification [6], a combination of multiwavelength range profiles and sequential evidence accumulation produces good familiarity discrimination, with the c-index approaching 1 as the number of sequential observations grows.

Fig. 1b also demonstrates the importance of the proper choice of familiarity measure. The jagged ROC curve was produced by a familiarity discrimination simulation identical to that which resulted in the 100-sequential-view smooth curve, but using the match function (2) instead of $\phi$ as given by (1).

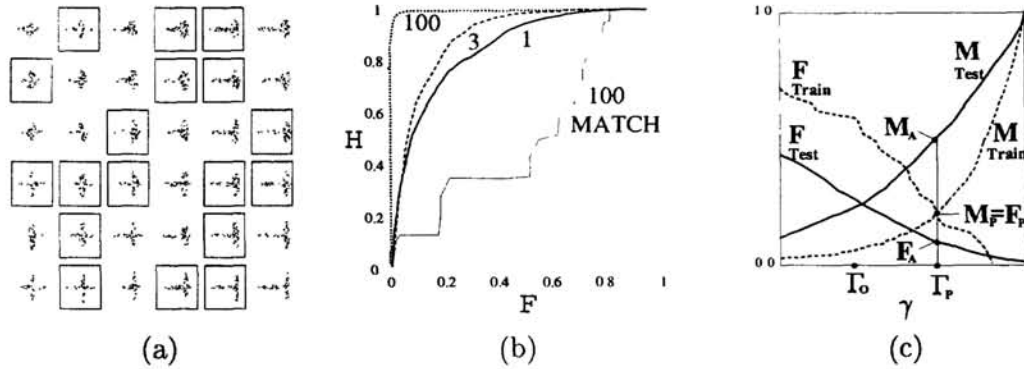

<center>(a)                                    (b)                                    (c)</center>

Figure 1:(a) 36 simulation targets with 6 wing positions and 6 wing lengths, and 100 scattering centers per target. Boxes indicate randomly selected familiar targets. (b) ROC curves from ARTMAP-FD simulations, with multiwavelength range profiles having 40 center frequencies. Sequential evidence accumulation for 1, 3 and 100 views uses familiarity measure (1) (smooth curves); and for 100 views uses the match function (2) (jagged curve). (c) Training and test curves of miss rate $M = (1 - H)$ and false alarm rate $F$ *vs* threshold $\gamma$, for 36 targets and one view. Training curves intersect at the point where $\gamma = \Gamma_P$ (predicted); and test curves intersect near the point where $\gamma = \Gamma_O$ (optimal). The training curves are based on data from the first training epoch, the test curves on data from 3 training epochs.

## 5  Familiarity threshold selection

When a system is placed in operation, one particular decision threshold $\gamma = \Gamma$ must be chosen. In a given application, selection of $\Gamma$ depends upon the relative cost of errors due to missed targets and false alarms. The optimal $\Gamma$ corresponds to a point on the parameterized ROC curve that is typically close to the upper left-hand corner of the unit square, to maximize correct selection of familiar targets ($H$) while minimizing incorrect selection of unfamiliar targets ($F$).

**Validation set method.** To determine a predicted threshold $\Gamma_P$, the training data is partitioned into a training subset and a validation subset. The network is trained on the training subset, and an ROC curve $(F(\gamma), H(\gamma))$ is calculated for the validation subset. $\Gamma_P$ is then taken to be the point on the curve that maximizes $[H(\gamma) - F(\gamma)]$. (For ease of computation the symmetry point on the curve, where $1 - H(\gamma) = F(\gamma)$, can yield a good approximation.) For a familiarity discrimination task the validation set must include examples of classes not present in the training set. Once $\Gamma_P$ is determined, the training subset and validation subset should be recombined and the network retrained on the complete training set. The retrained network and the predicted threshold $\Gamma_P$ are then employed for familiarity discrimination on the test set.

**On-line threshold determination.** During ARTMAP-FD training, category nodes compete for new patterns as they are presented. When a node $J$ wins the competition, learning expands the category hyperbox $R_J$ enough to enclose the training pattern **a**. The familiarity measure $\phi$ for each training set input then becomes equal to 1. However, *before* this learning takes place, $\phi$ can be less than 1, and the degree to which this initial value of $\phi$ is less than 1 reflects the distance from the training pattern to $R_J$. An event of this type—a training pattern successfully coded by a category node—is taken to be representative of familiar test-set patterns. The corresponding initial values of $\phi$ are thus used to generate a training

hit rate curve, where $H(\gamma)$ equals the fraction of training inputs with $\phi > \gamma$.

What about false alarms? By definition, all patterns presented during training are familiar. However, a reset event during training (Sec. 2) resembles the arrival of an unfamiliar pattern during testing. Recall that a reset occurs when a category node that predicts class $K$ wins the competition for a pattern that actually belongs to a different class $k$. The corresponding values of $\phi$ for these events can thus be used to generate a training false-alarm rate curve, where $F(\gamma)$ equals the fraction of match-tracking inputs with initial $\phi > \gamma$.

Predictive accuracy is improved by use of a reduced set of $\phi$ values in the training-set ROC curve construction process. Namely, training patterns that fall inside $R_J$, where $\phi = 1$, are not used because these exemplars tend to distort the miss rate curve. In addition, the *first* incorrect response to a training input is the best predictor of the network's response to an unfamiliar testing input, since sequential search will not be available during testing. Finally, giving more weight to events occurring later in the training process improves accuracy. This can be accomplished by first computing training curves $H(\gamma)$ and $F(\gamma)$ and a preliminary predicted threshold $\Gamma_P$ using the reduced training set; then recomputing the curves and $\Gamma_P$ from data presented only after the system has activated the final category node of the training process (Fig. 1c). The final predicted threshold $\Gamma_P$ averages these values. This calculation can still be made on-line, by taking the "final" node to be the last one activated.

Table 1 shows that applying on-line threshold determination to simulated radar range profile data gives good predictions for the actual hit and false alarm rates, $H_A$ and $F_A$. Furthermore, the $H_A$ and $F_A$ so obtained are close to optimal, particularly when the ROC curve has a c-index close to one. The method is effective even when testing involves sequential evidence accumulation, despite the fact that the training curves use only single views of each target.

## 6   NEN

Near-enough-neighbor (NEN) [7, 8] is a familiarity discrimination algorithm based on the single nearest neighbor classifier. For each familiar class $K$, the familiarity threshold $\Delta_K$ is the largest distance between any training pattern of class $K$ and its nearest neighbor also of class $K$. During testing, a test pattern is declared unfamiliar if the distance to its nearest neighbor is greater than the threshold $\Delta_K$ corresponding to the class $K$ of that nearest neighbor.

We have extended NEN to $k > 1$ by retaining the above definition of the $\Delta_K$'s, while taking the comparison during testing to be between $\Delta_K$ and the distance between the test pattern and the closest of its $k$ nearest neighbors which is of the class $K$ to which the test pattern is deemed to belong.

## 7   Radar pulse data

Identifying the type of emitter from which a radar signal was transmitted is an important task for radar electronic support measures (ESM) systems. Familiarity discrimination is a key component of this task, particularly as the continual proliferation of new emitters outstrips the ability of emitter libraries to document every sort of emitter which may be encountered.

The data analyzed here, gathered by Defense Research Establishment Ottawa, con-

|                 | 3x3 | | 6x6 | | 6x6* | |
|-----------------|--------|---------|--------|---------|--------|---------|
|                 | actual | optimal | actual | optimal | actual | optimal |
| hit rate        | 0.81   | 0.86    | 0.77   | 0.77    | 0.99   | 0.98    |
| false alarm rate| 0.11   | 0.14    | 0.24   | 0.23    | 0.06   | 0.02    |
| accuracy        | 0.95   | 1.00    | 0.93   | 1.00    | 1.00   | 1.00    |

Table 1: Familiarity discrimination, using ARTMAP-FD with on-line threshold prediction, of simulated radar range profile data. Training on half the target classes (boxed "aircraft" in Fig. 1a), testing on all target classes. (In 3x3 case, 4 classes out of 9 total used for training.) Accuracy equals the fraction of correctly-classified targets out of familiar targets selected by the network as familiar. The results for the 6x6* dataset involve sequential evidence accumulation, with 100 observations (5 sec.) per test target. Radar range profile simulations use 40 center frequencies evenly spaced between 18GHz and 22GHz, and $wp \times wl$ simulated targets, where $wp$ =number of wing positions and $wl$ =number of wing lengths. The number of range bins (2/3 m. per bin) is 60, so each pattern vector has (60 range bins) × (40 center frequencies) = 2400 components. Training patterns are at 21 evenly spaced aspects in a $10°$ angular range and, for each viewing angle, at 15 downrange shifts evenly spaced within a single bin width. Testing patterns are at random aspects and downrange shifts within the angular range and half the total range profile extent of (60 bins) × (2/3 m.) =40 m.

| method    | ARTMAP-FD | NEN | | | | | |
|-----------|-----------|-----------------------|-----------|------------|-----------|-----------|------------|
|           |           | city-block metric | | | Euclidean metric | | |
|           |           | $k=1$ | $k=5$ | $k=25$ | $k=1$ | $k=5$ | $k=25$ |
| hit rate  | 0.95      | 0.94  | 0.94  | 0.93   | 0.94  | 0.93  | 0.92   |
| f. a. rate| 0.02      | 0.13  | 0.04  | 0.02   | 0.14  | 0.05  | 0.02   |
| accuracy  | 1.00      | 1.00  | 1.00  | 1.00   | 0.99  | 1.00  | 1.00   |
| memory    | 21        | 446 | | | | | |

Table 2: Familiarity discrimination of radar pulse data set, using ARTMAP-FD and NEN with different metrics and values of $k$. Figure given for memory is twice number of $F_2$ nodes (due to complement coding) for ARTMAP-FD, number of training patterns for NEN. Training (single epoch) on first three quarters of data in classes 1-9, testing on other quarter of data in classes 1-9 and all data in classes 10-12. (Values given are averages over four cyclic permutations of the the 12 classes.) ARTMAP-FD familiarity threshold determined by validation-set method with retraining.

sist of radar pulses from 12 shipborne navigation radars [9]. Fifty pulses were collected from each radar, with the exception of radars #7 (100 pulses) and #8 (200 pulses). The pulses were preprocessed to yield 800 15-component vectors. with the components taking values between 0 and 1.

## 8   Results

From Table 2, ARTMAP-FD is seen to perform effective familiarity discrimination on the radar pulse data. NEN ($k = 1$) performs comparatively poorly. Extensions of NEN to $k > 1$ perform well. During fielded operation these would incur the cost of the additional computation required to find the $k$ nearest neighbors of the current test pattern, as well as the cost of higher memory requirements[1] relative to ARTMAP-FD. The combination of low hit rate with low false alarm rate obtained by NEN on the simulated radar range profile datasets (Table 3) suggests that the algorithm performs poorly here because it selects a familiarity threshold which is

| method | ARTMAP-FD | | NEN | | | | |
|---|---|---|---|---|---|---|---|
| | | | $k=1$ | $k=5$ | $k=99$ | $k=1$ | $k=5$ |
| dataset | 3x3 | 6x6 | 3x3 | | | 6x6 | |
| hit rate | 0.81 | 0.77 | 0.11 | 0.11 | 0.11 | 0.14 | 0.14 |
| false alarm rate | 0.11 | 0.24 | 0.00 | 0.00 | 0.00 | 0.00 | 0.00 |
| accuracy | 0.95 | 0.93 | 1.00 | 1.00 | 1.00 | 1.00 | 1.00 |
| memory | 12 | 88 | 1260 | | | 5670 | |

Table 3: Familiarity discrimination of simulated radar range profiles using ARTMAP-FD and NEN with different values of $k$. Training and testing as in Table 1. ARTMAP-FD familiarity threshold determined by on-line method. City-block metric used with NEN; results with Euclidean metric were slighlty poorer.

too high. ARTMAP-FD on-line threshold selection, on the other hand, yields a value for the familiarity threshold which balances the desiderata of high hit rate and low false alarm rate.

This research was supported in part by grants from the Office of Naval Research, ONR N00014-95-1-0657 (S. G.) and ONR N00014-96-1-0659 (M. A. R., W. W. S.), and by a grant from the Defense Advanced Research Projects Agency and the Office of Naval Research, ONR N00014-95-1-0409 (S. G., M. A. R., W. W. S.). E. G. was supported in part by the Defense Research Establishment Ottawa and the Natural Sciences and Engineering Research Council of Canada.

## Footnotes

[1]The memory requirements of $k$NN pattern classifiers can be reduced by editing techniques[8], but how the use of these methods affects performance of $k$NN-based familiarity discrimination methods is an open question.

# References

[1] Carpenter, G. A., Rubin, M. A., & Streilein, W. W., ARTMAP-FD: Familiarity discrimination applied to radar target recognition, in *ICNN'97: Proceedings of the IEEE International Conference on Neural Networks*, Houston, June 1997;

[2] Carpenter, G. A., Rubin, M. A., & Streilein, W. W., Threshold Determination for ARTMAP-FD Familiarity Discrimination, in C. H. Dagli *et al.*, eds., *Intelligent Engineering Systems Through Artificial Neural Networks*, **7**, 23-28, ASME, New York, 1997.

[3] Carpenter, G. A., Grossberg, S., Markuzon, N., Reynolds, J. H., & Rosen, D. B., Fuzzy ARTMAP: A neural network architecture for incremental supervised learning of analog multidimensional maps, *IEEE Transactions on Neural Networks*, **3**, 698-713, 1992.

[4] Carpenter, G. A., Grossberg, S., & Rosen. D. B., Fuzzy ART: Fast stable learning and categorization of analog patterns by an adaptive resonance system, *Neural Networks*, **4**, 759-771, 1991.

[5] Carpenter, G. A., & Ross, W. D., ART-EMAP: A neural network architecture for object recognition by evidence accumulation, *IEEE Transactions on Neural Networks*, **6**, 805-818, 1995.

[6] Rubin, M. A., Application of fuzzy ARTMAP and ART-EMAP to automatic target recognition using radar range profiles, *Neural Networks*, **8**, 1109-1116, 1995.

[7] Dasarathy, B. V., Is your nearest neighbor near enough a neighbor?, in Lainious, D. G. and Tzannes, N. S., eds. *Applications and Research in Informations Systems and Sciences*, **1**, 114-117, Hemisphere Publishing Corp., Washington, 1977.

[8] Dasarathy, B. V., ed., *Nearest Neighbor(NN) Norm: NN Pattern Classification Techniques*, IEEE Computer Society Press, Los Alamitos, CA, 1991.

[9] Granger, E., Savaria, Y., Lavoie, P., & Cantin, M.-A., A comparison of self-organizing neural networks for fast clustering of radar pulses, *Signal Processing*, **64**, 249-269, 1998.